# Second-order Learning Algorithm with Squared Penalty Term

**Kazumi Saito**      **Ryohei Nakano**
NTT Communication Science Laboratories
2 Hikaridai, Seika-cho, Soraku-gun, Kyoto 619-02 Japan
{saito,nakano}@cslab.kecl.ntt.jp

## Abstract

This paper compares three penalty terms with respect to the efficiency of supervised learning, by using first- and second-order learning algorithms. Our experiments showed that for a reasonably adequate penalty factor, the combination of the squared penalty term and the second-order learning algorithm drastically improves the convergence performance more than 20 times over the other combinations, at the same time bringing about a better generalization performance.

## 1   INTRODUCTION

It has been found empirically that adding some penalty term to an objective function in the learning of neural networks can lead to significant improvements in network generalization. Such terms have been proposed on the basis of several viewpoints such as weight-decay (Hinton, 1987), regularization (Poggio & Girosi, 1990), function-smoothing (Bishop, 1995), weight-pruning (Hanson & Pratt, 1989; Ishikawa, 1990), and Bayesian priors (MacKay, 1992; Williams, 1995). Some are calculated by using simple arithmetic operations, while others utilize higher-order derivatives. The most important evaluation criterion for these terms is how the generalization performance improves, but the learning efficiency is also an important criterion in large-scale practical problems; i.e., computationally demanding terms are hardly applicable to such problems. Here, it is naturally conceivable that the effects of penalty terms depend on learning algorithms; thus, we need comparative evaluations.

This paper evaluates the efficiency of first- and second-order learning algorithms

with three penalty terms. Section 2 explains the framework of the present learning and shows a second-order algorithm with the penalty terms. Section 3 shows experimental results for a regression problem, a graphical evaluation, and a penalty factor determination using cross-validation.

# 2   LEARNING WITH PENALTY TERM

## 2.1   Framework

Let $\{(\mathbf{x}_1, y_1), \cdots, (\mathbf{x}_m, y_m)\}$ be a set of examples, where $\mathbf{x}_t$ denotes an $n$-dimensional input vector and $y_t$ a target value corresponding to $\mathbf{x}_t$. In a three-layer neural network, let $h$ be the number of hidden units, $\mathbf{w}_j$ $(j = 1, \cdots, h)$ be the weight vector between all the input units and the hidden unit $j$, and $\mathbf{w}_0 = (w_{00}, \cdots, w_{0h})^T$ be the weight vector between all the hidden units and the output unit; $w_{j0}$ means a bias term and $x_{t0}$ is set to 1. Note that $\mathbf{a}^T$ denotes the transposed vector of $\mathbf{a}$. Hereafter, a vector consisting of all parameters, $(\mathbf{w}_0^T, \cdots, \mathbf{w}_h^T)^T$, is simply expressed as $\mathbf{\Phi} = (\phi_1, \cdots, \phi_N)^T$, where $N(= nh + 2h + 1)$ denotes the dimension of $\mathbf{\Phi}$. Then, the training error in the three-layer neural network can be defined as follows:

$$f(\mathbf{\Phi}) \;=\; \frac{1}{2} \sum_{t=1}^{m} \left\{ y_t - \left( w_{00} + \sum_{j=1}^{h} w_{0j} \sigma(\mathbf{w}_j^T \mathbf{x}_t) \right) \right\}^2, \tag{1}$$

where $\sigma(u)$ represents a sigmoidal function, $\sigma(u) = 1/(1 + e^{-u})$.

In this paper, we consider the following three penalty terms:

$$\Omega_1(\mathbf{\Phi}) = \frac{1}{2} \sum_{k=1}^{N} \phi_k^2, \quad \Omega_2(\mathbf{\Phi}) = \sum_{k=1}^{N} |\phi_k|, \quad \Omega_3(\mathbf{\Phi}) = \frac{1}{2} \sum_{k=1}^{N} \frac{\phi_k^2}{1 + \phi_k^2}. \tag{2}$$

Hereafter, $\Omega_1$, $\Omega_2$, and $\Omega_3$ are referred to as the squared (Hinton, 1987; MacKay, 1992), absolute (Ishikawa, 1990; Williams, 1995), and normalized (Hanson & Pratt, 1989) penalty terms, respectively. Then, learning with one of these terms can be defined as the problem of minimizing the following objective function

$$F_i(\mathbf{\Phi}) \;=\; f(\mathbf{\Phi}) + \mu \Omega_i(\mathbf{\Phi}), \tag{3}$$

where $\mu$ is a penalty factor.

## 2.2   Second-order Algorithm with Penalty Term

In order to minimize the objective function, we employ a newly invented second-order learning algorithm based on a quasi-Newton method, called BPQ (Saito & Nakano, 1997), where the descent direction, $\Delta\mathbf{\Phi}$, is calculated on the basis of a partial BFGS update and a reasonably accurate step-length, $\lambda$, is efficiently calculated as the minimal point of a second-order approximation. Here, the partial BFGS update can be directly applied, while the step-length $\lambda$ is evaluated as follows:

$$\lambda \;=\; \frac{-\nabla F_i(\mathbf{\Phi}) \Delta\mathbf{\Phi}^T}{\Delta\mathbf{\Phi}^T \nabla^2 F_i(\mathbf{\Phi}) \Delta\mathbf{\Phi}} \;=\; \frac{-\nabla F_i(\mathbf{\Phi}) \Delta\mathbf{\Phi}^T}{\Delta\mathbf{\Phi}^T \nabla^2 f(\mathbf{\Phi}) \Delta\mathbf{\Phi} + \mu \Delta\mathbf{\Phi}^T \nabla^2 \Omega_i(\mathbf{\Phi}) \Delta\mathbf{\Phi}}. \tag{4}$$

The quadratic form for the training error term, $\Delta\Phi^T\nabla^2 f(\Phi)\Delta\Phi$, can be calculated efficiently with the computational complexity of $Nm + O(hm)$ by using the procedure of BPQ, while those for penalty terms are calculated as follows:

$$\Delta\Phi^T\nabla^2\Omega_1(\Phi)\Delta\Phi = \sum_{k=1}^{N}\Delta\phi_k^2, \quad \Delta\Phi^T\nabla^2\Omega_2(\Phi)\Delta\Phi = 0,$$

$$\Delta\Phi^T\nabla^2\Omega_3(\Phi)\Delta\Phi = \sum_{k=1}^{N}\frac{(1 - 3\phi_k^2)\Delta\phi_k^2}{(1 + \phi_k^2)^3}. \tag{5}$$

Note that, in the step-length calculation, $\Delta\Phi^T\nabla^2 F_i(\Phi)\Delta\Phi$ is basically assumed to be positive. The three terms have a different effect on it, i.e., the squared penalty term always adds a non-negative value; the absolute penalty term has no effect; the normalized penalty term may add a negative value if many weight values are larger than $\sqrt{1/3}$. This indicates that the squared penalty term has a desirable feature. Incidentally, we can employ other second-order learning algorithms such as SCG (Møller, 1993) or OSS (Battiti, 1992), but BPQ worked the most efficiently among them in our own experience (Saito & Nakano, 1997).

## 3 EVALUATION BY EXPERIMENTS

### 3.1 Regression Problem

By using a regression problem for a function $y = (1 - x + 2x^2)e^{-0.5x^2}$, the learning performance of adding a penalty term was evaluated. In the experiment, a value of $x$ was randomly generated in the range of $[-4, 4]$, and the corresponding value of $y$ was calculated from $x$; each value of $y$ was corrupted by adding Gaussian noise with a mean of 0 and a standard deviation of 0.2. The total number of training examples was set to 30. The number of hidden units was set to 5, where the initial values for the weights between the input and hidden units were independently generated according to a normal distribution with a mean of 0 and a standard deviation of 1; the initial values for the weights between the hidden and output units were set to 0, but the bias value at the output unit was initially set to the average output value of all training examples. The iteration was terminated when the gradient vector was sufficiently small (i.e., $\|\nabla F_i(\Phi)\|^2/N < 10^{-12}$) or the total processing time exceeded 100 seconds. The penalty factor $\mu$ was changed from $2^0$ to $2^{-19}$ by multiplying by $2^{-1}$; trials were performed 20 times for each penalty factor.

Figure 1 shows the training examples, the true function, and a function obtained after learning without a penalty term. We can see that such a learning over-fitted the training examples to some degree.

### 3.2 Evaluation using Second-order Algorithm

By using BPQ, an evaluation was made after adding each penalty term. Figure 2(a) compares the generalization performance, which was evaluated by using the average RMSE (root mean squared error) for a set of $5,000$ test examples. The best possible RMSE level is 0.2 because each test example includes the same amount of Gaussian noise given to each training example. For each penalty term, the generalization performance was improved when $\mu$ was set adequately, but the normalized

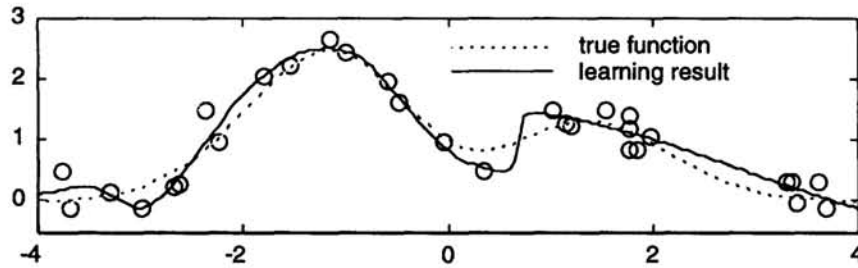

Figure 1: Learning problem

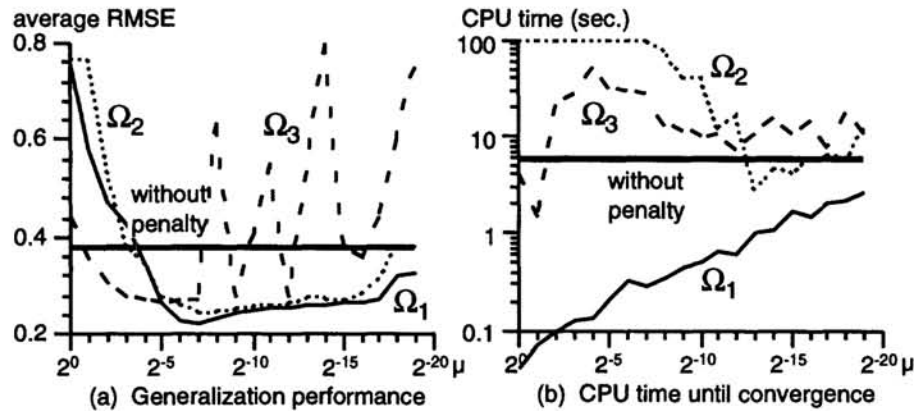

Figure 2: Comparison using second-order algorithm BPQ

penalty term was the most unstable among the three, because it frequently got stuck in undesirable local minima. Figure 2(b) compares the processing time[1] until convergence. In comparison to the learning without a penalty term, the squared penalty term drastically decreased the processing time especially when $\mu$ was large, while the absolute penalty term did not converge when $\mu$ was large; the normalized penalty term generally required a larger processing time. Thus, only the squared penalty term improved the convergence performance more than $2 \sim 100$ times, keeping a better generalization performance for an adequate penalty factor.

### 3.3  Evaluation using First-order Algorithm

By using BP, a similar evaluation was made after adding each penalty term. Here, we adopted Silva and Almeida's learning rate adaptation rule (Silva & Almeida, 1990), i.e., learning rate $\eta_k$ for each weight $\phi_k$ is adjusted by the signs of two successive gradient values[2]. Figure 3(a) compares the generalization performance and Figure 3(b) compares the processing time until convergence, where the average processing time for the trials without a penalty term is not displayed because all trials did not converge within 100 seconds. For each penalty term, the generalization

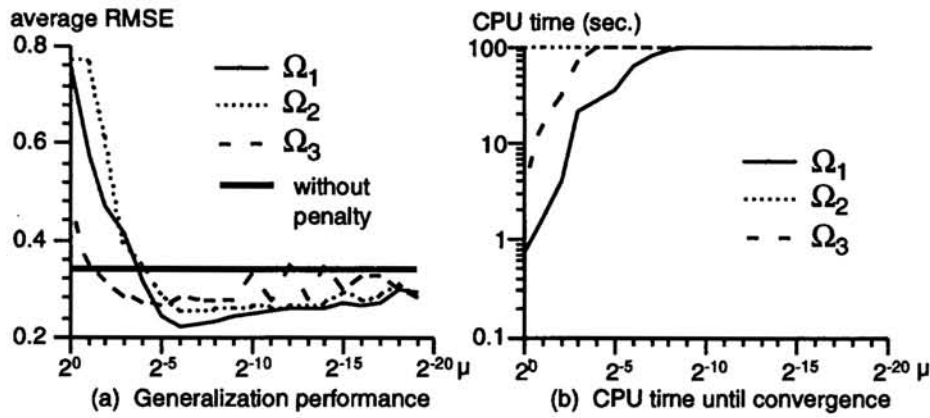

Figure 3: Comparison using first-order algorithm BP

performance was improved when $\mu$ was set adequately. Note that BP with the squared penalty term $\Omega_1$ required more processing time than BPQ with $\Omega_1$. As for the normalized penalty term $\Omega_3$, BP with $\Omega_3$ worked more stably than BPQ with $\Omega_3$. Incidentally, the generalization performance of BP without a penalty term was better than that of BPQ without it; we predict that this is because the effect of early stopping (Bishop, 1995) worked for BP. Actually, for the training examples, the average RMSE of BP without a penalty term was 0.138, while that of BPQ without it was 0.133.

## 3.4 Graphical Evaluation

In order to graphically examine the reasons why the effect of the addition of each penalty term differed, we designed a simple problem; that is, learning a function $y = \sigma(w_1 x) + \sigma(w_2 x)$, where only two weights, $w_1$ and $w_2$, are adjustable. In the three-layer network, the input and output layers consist of only one unit, while the hidden layer consists of two units. Note that the weights between the hidden units and the output unit are fixed at 1, there is no bias, and the activation function of hidden units is assumed to be $\sigma(x) = 1/(1 + \exp(-x))$. Each target value $y_t$ was calculated from the corresponding input value $x_t \in \{-0.2, -0.1, 0, 0.1, 0.2\}$ by setting $(w_1, w_2) = (1, 3)$.

Figure 4 shows the learning trajectories on error contour maps with respect to $w_1$ and $w_2$ during 100 iterations starting at $(w_1, w_2) = (-1, -3)$, where the penalty factor $\mu$ was set to 0.1 or 0.01. Here, BPQ was used as a learning algorithm. The contours for the squared penalty term form ovals, making BPQ learn easily. When $\mu = 0.1$, the contours for the absolute penalty term form an almost square-like shape, and the learning trajectories oscillate near the origin ($w_1 = w_2 = 0$), due to the discontinuity of the gradient function. The contours for the normalized penalty term form a valley, making BPQ's learning more difficult.

## 3.5 Determining Penalty Factor

In general, for a given problem, we cannot know an adequate penalty factor in advance. Given a limited number of examples, we must find a reasonably adequate

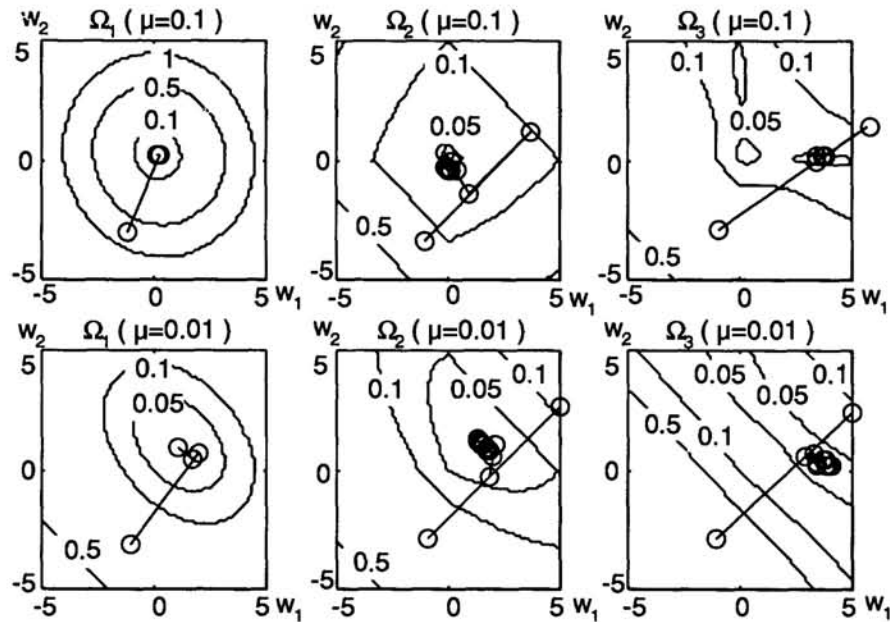

Figure 4: Graphical evaluation

penalty factor. The procedure of cross-validation (Stone, 1978) is adopted for this purpose. Since we knew the combination of the squared penalty term $\Omega_1$ and the second-order algorithm BPQ works very efficiently, we performed experiments using the above regression problem with exactly the same experimental conditions.

Figure 5 shows the experimental results, where the procedure of cross-validation was implemented as a leave-one-out method, and the initial weight values for evaluating the cross-validation error were set as the learning results of the entire training examples. Figure 5(a) compares the average generalization error and the average cross-validation error. Although the cross-validation error was a pessimistic estimator of the generalization error, it showed the same tendency and was minimized at almost the same penalty factor. Figure 5(b) shows the average processing time and its standard deviation; although the processing time includes the cross-validation evaluation, we can see that the learning was performed quite efficiently.

## 4   CONCLUSION

This paper investigated the efficiency of supervised learning with each of three penalty terms, by using first- and second-order learning algorithms, BP and BPQ. Our experiments showed that for a reasonably adequate penalty factor, the combination of the squared penalty term and the second-order algorithm drastically improves the convergence performance about 20 times over the other combinations, together with an improvement in the generalization performance. In the case of other second-order learning algorithms such as SCG or OSS, similar results are possible because the main difference between BPQ and those other algorithms involves only the learning efficiency. In the future, we plan to do further evaluations using larger-scale problems.

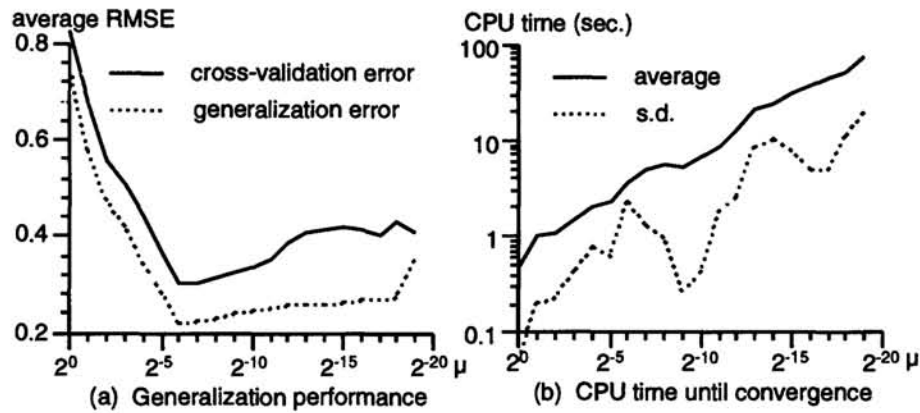

Figure 5: Learning result

## Footnotes

[1]Our experiments were done on SUN S-4/20 computers.

[2]The increasing and decreasing parameters were set to 1.1 and 1/1.1, respectively, as recommended by (Silva & Almeida, 1990); if the value of the objective function increases, all learning rates are halved until the value decreases.

## References

Battiti, R. (1992) First- and second-order methods for learning between steepest descent and Newton's method. *Neural Computation* **4**(2):141–166.

Bishop, C.M. (1995) *Neural networks for pattern recognition.* Clarendon Press.

Hanson, S.J. & Pratt, L. Y. (1989) Comparing biases for minimal network construction with back-propagation. In D. S. Touretzky (ed.), *Advances in Neural Processing Systems*, Volume 1, pp. 177–185. San Mateo, CA: Morgan Kaufmann.

Hinton, G.E. (1987) Learning translation invariant recognition in massively parallel networks. In J. W. de Bakker, A. J. Nijman and P. C. Treleaven (eds.), *Proceedings PARLE Conference on Parallel Architectures and Languages Europe*, pp. 1–13. Berlin: Springer-Verlag.

Ishikawa, M. (1990) A structural learning algorithm with forgetting of link weight. Tech. Rep. TR-90-7, Electrotechnical Lab. Tsukuba-City, Japan.

MacKay, D.J.C. (1992) Bayesian interpolation. *Neural Computation* **4**(3):415–447.

Møller, M.F. (1993) A scaled conjugate gradient algorithm for fast supervised learning. *Neural Networks* **6**(4):525–533.

Poggio, T. & Girosi, F. (1990) Regularization algorithms for learning that are equivalent to multilayer networks. *Science* **247**:978–982.

Saito, K. & Nakano, R. (1997) Partial BFGS update and efficient step-length calculation for three-layer neural networks. *Neural Computation* **9**(1):239–257 (in press).

Silva, F.M. & Almeida, L.B. (1990) Speeding up backpropagation. In R. Eckmiller (ed.), *Advanced Neural Computers*, pp. 151–160. Amsterdam: North–Holland.

Stone, M. (1978) Cross-validation: A review. *Operationsforsch. Statist. Ser. Statistics B* **9**(1):111–147.

Williams, P.M. (1995) Bayesian regularization and pruning using a Laplace prior. *Neural Computation* **7**(1):117–143.
